# Training Knowledge-Based Neural Networks to Recognize Genes in DNA Sequences

**Michiel O. Noordewier**
Computer Science
Rutgers University
New Brunswick, NJ 08903

**Geoffrey G. Towell**
Computer Sciences
University of Wisconsin
Madison, WI 53706

**Jude W. Shavlik**
Computer Sciences
University of Wisconsin
Madison, WI 53706

## Abstract

We describe the application of a hybrid symbolic/connectionist machine learning algorithm to the task of recognizing important genetic sequences. The symbolic portion of the KBANN system utilizes inference rules that provide a roughly-correct method for recognizing a class of DNA sequences known as *eukaryotic splice-junctions*. We then map this "domain theory" into a neural network and provide training examples. Using the samples, the neural network's learning algorithm adjusts the domain theory so that it properly classifies these DNA sequences. Our procedure constitutes a general method for incorporating preexisting knowledge into artificial neural networks. We present an experiment in molecular genetics that demonstrates the value of doing so.

## 1  Introduction

Often one has some preconceived notions about how to perform some classification task. It would be useful to incorporate this knowledge into a neural network, and then use some training examples to refine these approximately-correct rules of thumb. This paper describes the KBANN *(Knowledge-Based Artificial Neural Networks)* hybrid learning system and demonstrates its ability to learn in the complex domain of molecular genetics. Briefly, KBANN uses a knowledge base of hierarchically-structured rules (which may be both incomplete and incorrect) to form an artificial neural network (ANN). In so doing, KBANN makes it possible to apply neural learning techniques to the empirical improvement of knowledge bases.

The task to be learned is the recognition of certain DNA (deoxyribonucleic acid) subsequences important in the expression of genes. A large governmental research

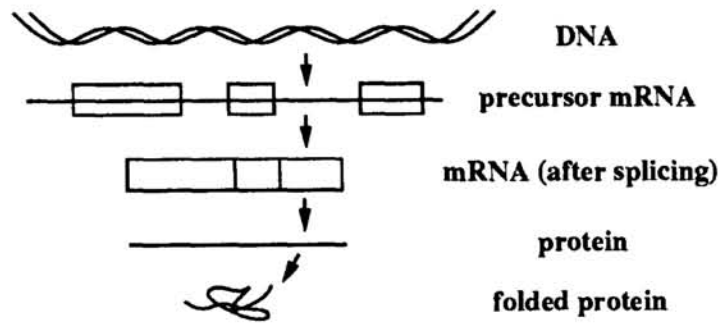

Figure 1: Steps in the Expression of Genes

program, called the Human Genome Initiative, has recently been undertaken to determine the sequence of DNA in humans, estimated to be $3 \times 10^9$ characters of information. This provides a strong impetus to develop genetic-analysis techniques based solely on the information contained in the sequence, rather than in combination with other chemical, physical, or genetic techniques. DNA contains the information by which a cell constructs protein molecules. The cellular expression of proteins proceeds by the creation of a "message" ribonucleic acid (mRNA) copy from the DNA template (Figure 1). This mRNA is then translated into a protein. One of the most unexpected findings in molecular biology is that large pieces of the mRNA are removed before it is translated further [1].

The utilized sequences (represented by boxes in Figure 1) are known as "exons", while the removed sequences are known as "introns", or intervening sequences. Since the discovery of such "split genes" over a decade ago, the nature of the splicing event has been the subject of intense research. The points at which DNA is removed (the boundaries of the boxes in Figure 1) are known as *splice-junctions*. The splice-junctions of eukaryotic[1] mRNA precursors contain patterns similar to those in Figure 2.

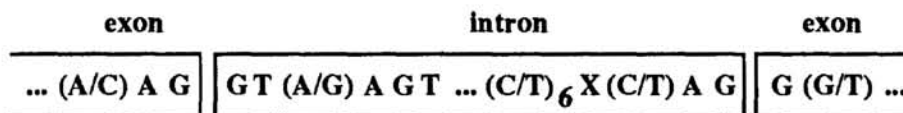

Figure 2: Canonical Splice-Junctions
DNA is represented by a string of characters from the set {A,G,C,T}.
In this figure, X represents any character, slashes represent disjunctive options, and subscripts indicate repetitions of a pattern.

However, numerous other locations can resemble these canonical patterns. As a result, these patterns do not by themselves reliably imply the presence of a splice-junction. Evidently, if junctions are to be recognized on the basis of sequence information alone, longer-range sequence information will have to be included in

the decision-making criteria. A central problem is therefore to determine the extent to which sequences surrounding splice-junctions differ from sequences surrounding spurious analogues.

We have recently described a method [9, 12] that combines empirical and symbolic learning algorithms to recognize another class of genetic sequences known as *bacterial promoters*. Our hybrid KBANN system was demonstrated to be superior to other empirical learning systems including decision trees and nearest-neighbor algorithms. In addition, it was shown to more accurately classify promoters than the methods currently reported in the biological literature. In this manuscript we describe the application of KBANN to the recognition of splice-junctions, and show that it significantly increases generalization ability when compared to randomly-initialized, single-hidden-layer networks (i.e., networks configured in the "usual" way). The paper concludes with a discussion of related research and the areas which our research is currently pursuing.

## 2    The KBANN Algorithm

KBANN uses a knowledge base of domain-specific inference rules in the form of PROLOG-like clauses to define what is initially known about a topic. The knowledge base need be neither complete nor correct; it need only support approximately correct reasoning. KBANN translates knowledge bases into ANNs in which units and links correspond to parts of knowledge bases. A detailed explanation of the procedure used by KBANN to translate rules into an ANN can be found in [12].

As an example of the KBANN method, consider the artificial knowledge base in Figure 3a which defines membership in category A. Figure 3b represents the hierarchical structure of these rules: solid and dotted lines represent *necessary* and *prohibitory* dependencies, respectively. Figure 3c represents the ANN that results from the translation into a neural network of this knowledge base. Units X and Y in Figure 3c are introduced into the ANN to handle the disjunction in the knowledge base. Otherwise, units in the ANN correspond to consequents or antecedents in the knowledge base. The thick lines in Figure 3c represent the links in the ANN that correspond to dependencies in the explanation. The weight on thick solid lines is 3, while the weight on thick dotted lines is -3. The lighter solid lines represent the links added to the network to allow refinement of the initial rules. At present, KBANN is restricted to non-recursive, propositional (i.e., variable-free) sets of rules.

Numbers beside the unit names in Figure 3c are the biases of the units. These biases are set so that the unit is active if and only if the corresponding consequent in the knowledge base is true.

As this example illustrates, the use of KBANN to initialize ANNs has two principle benefits. First, it indicates the features believed to be important to an example's classification. Second, it specifies important derived features; through their deduction the complexity of an ANN's final decision is reduced.

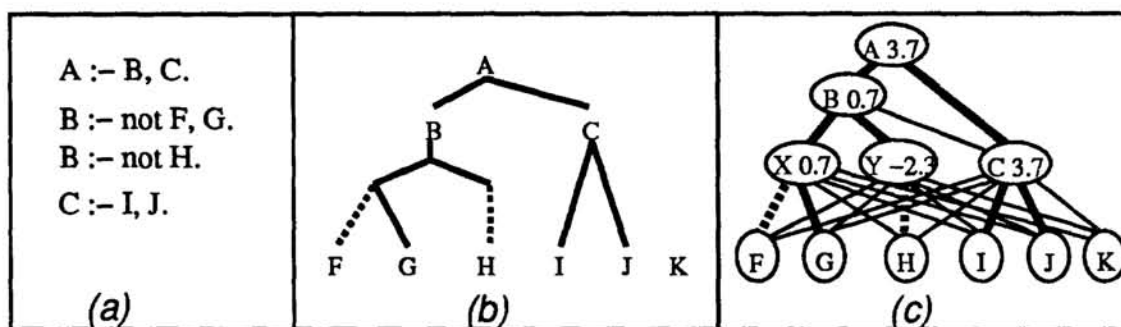

A :- B, C.
B :- not F, G.
B :- not H.
C :- I, J.

(a)          (b)          (c)

Figure 3: Translation of a Knowledge Base into an ANN

## 3   Problem Definition

The splice-junction problem is to determine into which of the following three categories a specified location in a DNA sequence falls: (1) exon/intron borders, referred to as *donors*, (2) intron/exon borders, referred to as *acceptors*, and (3) *neither*. To address this problem we provide KBANN with two sets of information: a set of DNA sequences 60 nucleotides long that are classified as to the category membership of their center and a domain theory that describes when the center of a sequence corresponds to one of these three categories.

Table 1 contains the initial domain theory used in the splice-junction recognition task. A special notation is used to specify locations in the DNA sequence. When a rule's antecedents refer to input features, they first state a relative location in the sequence vector, then the DNA symbol that must occur (e.g., @3=A). Positions are numbered negatively or positively depending on whether they occur before or after the possible junction location. By biological convention, position numbers of zero are not used. The set of rules was derived in a straightforward fashion from the biological literature [13]. Briefly, these rules state that a donor or acceptor sequence is present if characters from the canonical sequence (Figure 2) are present and triplets known as *stop codons* are absent in the appropriate positions.

The examples were obtained by taking the documented split genes from all primate gene entries in Genbank release 64.1 [?] that are described as complete. Each training example consists of a window that covers 30 nucleotides before and after each donor and acceptor site. This procedure resulted in 751 examples of acceptor and 745 examples of donors. Negative examples are derived from similarly-sized windows, which did not cross an intron/exon boundary, sampled at random from these sequences. Note that this differs from the usual practice of generating random sequences with base-frequency composition the same as the positive instances. However, we feel that this provides a more realistic training set, since DNA is known to be highly non-random [3]. Although many more negative examples were available, we used approximately as many negative examples are there were both donor and acceptors. Thus, the total data set we used had 3190 examples.

The network created by KBANN for the splice-junction problem has one output

Table 1: Knowledge Base for Splice-Junctions

```
donor :- @-3=M, @-2=A, @-1=G, @1=G, @2=T, @3=R,
         @4=A, @5=G, @6=T, not(don-stop).
don-stop :- @-3=T, @-2=A, @-1=A.      don-stop :- @-4=T, @-3=A, @-2=G.
don-stop :- @-3=T, @-2=A, @-1=G.      don-stop :- @-4=T, @-3=G, @-2=A.
don-stop :- @-3=T, @-2=G, @-1=A.      don-stop :- @-5=T, @-4=A, @-3=A.
don-stop :- @-4=T, @-3=A, @-2=A.      don-stop :- @-5=T, @-4=A, @-3=G.
don-stop :- @-5=T, @-4=G, @-3=A.
acceptor :- pyr-rich, @-3=Y, @-2=A, @-1=G, @1=G, @2=K, not(acc-stop).
pyr-rich :- 6 of (@-15=Y, @-14=Y, @-13=Y, @-12=Y, @-11=Y,
                  @-10=Y, @-9=Y, @-8=Y, @-7=Y, @-6=Y.)
acc-stop :- @1=T, @2=A, @3=A.         acc-stop :- @2=T, @3=A, @4=A.
acc-stop :- @1=T, @2=A, @3=G.         acc-stop :- @2=T, @3=A, @4=G.
acc-stop :- @1=T, @2=G, @3=A.         acc-stop :- @2=T, @3=G, @4=A.
acc-stop :- @3=T, @4=A, @5=A.         acc-stop :- @3=T, @4=A, @5=G.
acc-stop :- @3=T, @4=G, @5=A.
R :- A.  R :- G.  Y :- C.  Y :- T.  M :- C.  M :- A.  K :- G.  K :- T
```

units for each category to be learned; and four input units for each nucleotide in the DNA training sequences, one for each of the four values in the DNA alphabet. In addition, the rules for *acc-stop, don-stop, R, Y*, and *M* are considered definitional. Thus, the weights on the links and biases into these units were frozen. Also, the second rule only requires that six of its 11 antecedents be true. Finally, there are no rules in Table 1 for recognizing negative examples. So we added four unassigned hidden units and connected them to all of the inputs and to the output for the *neither* category. The final result is that the network created by KBANN has 286 units: 3 output units, 240 input units, 31 fixed-weight hidden units, and 12 tunable hidden units.

## 4   Experimental Results

Figure 4 contains a learning curve plotting the percentage of errors made on a set of "testing" examples by KBANN-initialized networks, as a function of the number of training examples. Training examples were obtained by randomly selecting examples from the population of 3190 examples described above. Testing examples consisted of all examples in the population that were not used for training. Each data point represents the average of 20 repetitions of this procedure.

For comparison, the error rate for a randomly-initialized, fully-connected, two-layer ANN with 24 hidden units is also plotted in Figure 4. (This curve is expected to have an error rate of 67% for zero training examples. Test results were slightly better due to statistical fluctuations.) Clearly, the KBANN-initialized networks learned faster than randomly-initialized ANNs, making less than half the errors of the randomly-initialized ANNs when there were 100 or fewer training examples. However, when

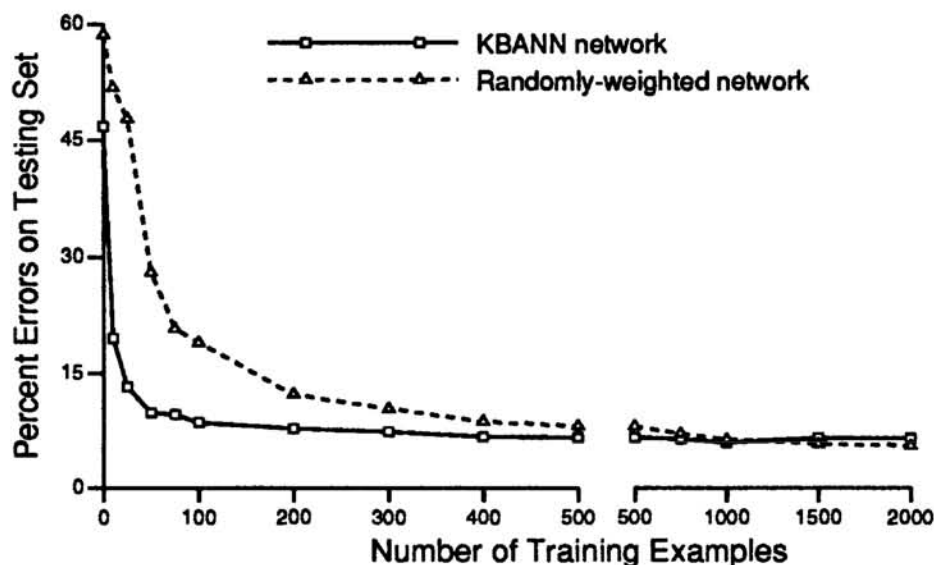

Figure 4: Learning Curve for Splice Junctions

large numbers of training examples were provided the randomly-initialized ANNs had a slightly lower error rate (5.5% vs. 6.4% for KBANN ). All of the differences in the figure are statistically significant.

## 5  Related and Future Research

Several others have investigated predicting splice-junctions. Staden [10] has devised a weight-matrix method that uses a perceptron-like algorithm to find a weighting function that discriminates two sets (true and false) of boundary patterns in known sequences. Nakata et al. [7] employ a combination of methods to distinguish between exons and introns, including Fickett's statistical method [5]. When applied to human sequences in the Genbank database; this approach correctly identified 81% of true splice-junctions. Finally, Lapedes et al. [6] also applied neural networks and decision-tree builders to the splice-junction task. They reported neural-network accuracies of 92% and claimed their neural-network approach performed significantly better than the other approaches in the literature at that time. The accuracy we report in this paper represents an improvement over these results. However, it should be noted that these experiments were not all performed under the same conditions.

One weakness of neural networks is that it is hard to understand what they have learned. We are investigating methods for the automatic translation into symbolic rules of trained KBANN-initialized networks [11]. These techniques take advantage of the human-comprehensible starting configuration of KBANN's networks to create a small set of hierarchically-structured rules that accurately reflect what the network learned during training. We are also currently investigating the use of richer splice-junction domain theories, which we hope will improve KBANN's accuracy.

## 6   Conclusion

The KBANN approach allows ANNs to refine preexisting knowledge, generating ANN topologies that are well-suited to the task they are intended to learn. KBANN does this by using a knowledge base of approximately correct, domain-specific rules to determine the ANN's structure and initial weights. This provides an alternative to techniques that either shrink [2] or grow [4] networks to the "right" size. Our experiments on splice-junctions, and previously on bacterial promoters, [12] demonstrate that the KBANN approach can substantially reduce the number of training examples needed to reach a given level of accuracy on future examples.

This research was partially supported by Office of Naval Research Grant N00014-90-J-1941, National Science Foundation Grant IRI-9002413, and Department of Energy Grant DE-FG02-91ER61129.

## Footnotes

[1]Eukaryotic cells contain nuclei, unlike prokaryotic cells such as bacterial and viruses.

## References

[1] R. J. Breathnach, J. L. Mandel, and P. Chambon. Ovalbumin gene is split in chicken DNA. *Nature*, 270:314–319, 1977.

[2] Y. Le Cun, J. Denker, and S. Solla. Optimal brain damage. *Advances in Neural Information Processing Systems 2*, pages 598–605, 1990.

[3] G. Dykes, R. Bambara, K. Marians, and R. Wu. On the statistical significance of primary structural features found in DNA-protein interaction sites. *Nucleic Acids Research*, 2:327–345, 1975.

[4] S. Fahlman and C. Lebiere. The cascade-correlation learning architecture. *Advances in Neural Information Processing Systems 2*, pages 524–532, 1990.

[5] J. W. Fickett. Recognition of protein coding regions in DNA sequences. *Nucleic Acids Research*, 10:5303–5318, 1982.

[6] A. Lapedes, D. Barnes, C. Burks, R. Farber, and K. Sirotkin. Application of neural networks and other machine learning algorithms to DNA sequence analysis. In *Computers and DNA*, pages 157–182. Addison-Wesley, 1989.

[7] K. Nakata, M. Kanehisa, and C. DeLisi. Prediction of splice junctions in mrna sequences. *Nucleic Acids Research*, 13:5327–5340, 1985.

[8] M. C. O'Neill. Escherichia coli promoters: I. Consensus as it relates to spacing class, specificity, repeat substructure, and three dimensional orgainzation. *Journal of Biological Chemistry*, 264:5522–5530, 1989.

[9] J. W. Shavlik and G. G. Towell. An approach to combining explanation-based and neural learning algorithms. *Connection Science*, 1:233–255, 1989.

[10] R. Staden. Computer methods to locate signals in DNA sequences. *Nucleic Acids Research*, 12:505–519, 1984.

[11] G. G. Towell, M. Craven, and J. W. Shavlik. Automated interpretation of knowledge based neural networks. Technical report, University of Wisconsin, Computer Sciences Department, Madison, WI, 1991.

[12] G. G. Towell, J. W. Shavlik, and M. O. Noordewier. Refinement of approximately correct domain theories by knowledge-based neural networks. In *Proc. of the Eighth National Conf. on Artificial Intelligence*, pages 861–866, Boston, MA, 1990.

[13] J. D. Watson, N. H. Hopkins, J. W. Roberts, J. A. Steitz, and A. M. Weiner. *Molecular Biology of the Gene*, pages 634–647, 1987.